# Sequence and Tree Kernels
# with Statistical Feature Mining

**Jun Suzuki and Hideki Isozaki**
NTT Communication Science Laboratories, NTT Corp.
2-4 Hikaridai, Seika-cho, Soraku-gun, Kyoto,619-0237 Japan
{jun, isozaki}@cslab.kecl.ntt.co.jp

## Abstract

This paper proposes a new approach to feature selection based on a statistical feature mining technique for sequence and tree kernels. Since natural language data take discrete structures, convolution kernels, such as sequence and tree kernels, are advantageous for both the concept and accuracy of many *natural language processing* tasks. However, experiments have shown that the best results can only be achieved when limited small sub-structures are dealt with by these kernels. This paper discusses this issue of convolution kernels and then proposes a statistical feature selection that enable us to use larger sub-structures effectively. The proposed method, in order to execute efficiently, can be embedded into an original kernel calculation process by using *sub-structure mining* algorithms. Experiments on real NLP tasks confirm the problem in the conventional method and compare the performance of a conventional method to that of the proposed method.

## 1 Introduction

Since natural language data take the form of sequences of words and are generally analyzed into discrete structures, such as trees (parsed trees), *discrete kernels*, such as sequence kernels [7, 1] and tree kernels [2, 5], have been shown to offer excellent results in the *natural language processing (NLP)* field. Conceptually, these proposed kernels are defined as instances of *convolution kernels* [3, 11], which provides the concept of kernels over discrete structures.

However, unfortunately, experiments have shown that in some cases there is a critical issue with convolution kernels in NLP tasks [2, 1, 10]. That is, since natural language data contain many types of symbols, NLP tasks usually deal with extremely high dimension and sparse feature space. As a result, the convolution kernel approach can never be trained effectively, and it behaves like a nearest neighbor rule. To avoid this issue, we generally eliminate large sub-structures from the set of features used. However, the main reason for using convolution kernels is that we aim to use structural features easily and efficiently. If their use is limited to only very small structures, this negates the advantages of using convolution kernels.

This paper discusses this issue of convolution kernels, in particular sequence and tree ker-

nels, and proposes a new method based on statistical significant test. The proposed method deals only with those features that are statistically significant for solving the target task, and large significant sub-structures can be used without over-fitting. Moreover, by using *sub-structure mining* algorithms, the proposed method can be executed efficiently by embedding it in an original kernel calculation process, which is defined by the *dynamic-programming* (DP) based calculation.

## 2 Convolution Kernels for Sequences and Trees

Convolution kernels have been proposed as a concept of kernels for discrete structures, such as sequences, trees and graphs. This framework defines the kernel function between input objects as the convolution of "sub-kernels", i.e. the kernels for the decompositions (parts or sub-structures) of the objects. Let $X$ and $Y$ be discrete objects. Conceptually, convolution kernels $K(X, Y)$ enumerate all sub-structures occurring in $X$ and $Y$ and then calculate their inner product, which is simply written as: $K(X, Y) = \langle \phi(X), \phi(Y) \rangle = \sum_i \phi_i(X) \cdot \phi_i(Y)$. $\phi$ represents the feature mapping from the discrete object to the feature space; that is, $\phi(X) = (\phi_1(X), \ldots, \phi_i(X), \ldots)$. Therefore, with sequence kernels, input objects $X$ and $Y$ are sequences, and $\phi_i(X)$ is a sub-sequence; with tree kernels, $X$ and $Y$ are trees, and $\phi_i(X)$ is a sub-tree. Up to now, many kinds of sequence and tree kernels have been proposed for a variety of different tasks. To clarify the discussion, this paper basically follows the framework of [1], which proposed a *gapped word sequence kernel*, and [5], which introduced a *labeled ordered tree kernel*.

We can treat that sequence is one of the special form of trees if we say sequences are rooted by their last symbol and each node has one child each of a previous symbol. Thus, in this paper, the word 'tree' is always including sequence. Let $\mathcal{L}$ be a set of finite symbols. Then, let $\mathcal{L}^n$ be a set of symbols whose sizes are $n$ and $P(\mathcal{L}^n)$ be a set of trees that are constructed by $\mathcal{L}^n$. The meaning of "size" in this paper is the the number of nodes in a tree. We denote a tree $u \in P(\mathcal{L}_1^n)$ whose size is $n$ or less, where $\cup_{m=1}^n \mathcal{L}^m = \mathcal{L}_1^n$. Let $T$ be a tree and $\mathrm{sub}(T)$ be a function that returns a set of all possible sub-trees in $T$. We define a function $C_u(t)$ that returns a constant, $\lambda(0 < \lambda \leq 1)$, if the sub-tree $t$ covers $u$ with the same root symbol. For example, a sub-tree 'a-b-c-d', where 'a', 'b', 'c' and 'd' represent symbols and '-' represents an edge between symbols, covers sub-trees 'd', 'a-c-d' and 'b-d'. That is, $C_u(t) = \lambda$ if $u$ matches $t$ allowing the node skip, 0 otherwise. We also define a function $\gamma_u(t)$ that returns the difference of size of sub-trees $t$ and $u$. For example, if $t = $ a-b-c-d and $u = $ a-b, then $\gamma_u(t) = |4 - 2| = 2$.

Formally, sequence and tree kernels can be defined as the same form as

$$K^{\text{SK,TK}}(T^1, T^2) = \sum_{u \in P(\mathcal{L}_1^n)} \sum_{t^1 \in \mathrm{sub}(T^1)} C_u(t^1)^{\gamma_u(t^1)} \sum_{t^2 \in \mathrm{sub}(T^2)} C_u(t^2)^{\gamma_u(t^2)}. \quad (1)$$

Note that this formula is also including the node skip framework that is generally introduced only in sequence kernels[7, 1]; $\lambda$ is the decay factor that handles the gap present in sub-trees $u$ and $t$.

Sequence and tree kernels are defined in recursive formula to calculate them efficiently instead of the explicit calculation of Equation (1). Moreover, when implemented, these kernels can calculated in $O(n|T^1||T^2|)$, where $|T|$ represents the number of nodes in $T$, by using the DP technique. Note, that if the kernel does not use size restriction, the calculation cost becomes $O(|T^1||T^2|)$.

## 3   Problem of Applying Convolution Kernels to Real tasks

According to the original definition of convolution kernels, all of the sub-structures are enumerated and calculated for the kernels. The number of sub-structures in the input object usually becomes exponential against input object size. The number of symbols, $|\mathcal{L}|$, is generally very large number (i.e. more than 10,000) since words are treated as symbols. Moreover, the appearance of sub-structures (sub-sequences and sub-trees) are highly correlated with that of sub-structures of sub-structures themselves. As a result, the dimension of feature space becomes extremely high, and all kernel values $K(X, Y)$ are very small compared to the kernel value of the object itself, $K(X, X)$. In this situation, the convolution kernel approach can never be trained effectively, and it will behave like a nearest neighbor rule; we obtain a result that is very precise but with very low recall. The details of this issue were described in [2].

To avoid this, most conventional methods use an approach that involves smoothing the kernel values or eliminating features based on the sub-structure size. For sequence kernels, [1] use a feature elimination method based on the size of sub-sequence $n$. This means that the kernel calculation deals only with those sub-sequences whose length is $n$ or less. As well as the sequence kernel, [2] proposed a method that restricts the features based on sub-tree depth for tree kernels. These methods seem to work well on the surface, however, good results can only be achieved when $n$ is very small, i.e. $n = 2$ or 3. For example, $n = 3$ showed the best performance for parsing in the experimental results of [2], and $n = 2$ showed the best for the text classification task in [1]. The main reason for using these kernels is that they allow us to employ structural features simply and efficiently. When only small-sized sub-structures are used (i.e. $n = 2$ or 3), the full benefits of the kernels are missed.

Moreover, these results do not mean that no larger-sized sub-structures are useful. In some cases we already know that certain larger sub-structures can be significant features for solving the target problem. That is, significant larger sub-structures, which the conventional methods cannot deal with efficiently, should have the possibility of further improving the performance. The aim of the work described in this paper is to be able to use any significant sub-structure efficiently, regardless of its size, to better solve NLP tasks.

## 4   Statistical Feature Mining Method for Sequence and Tree Kernels

This section proposes a new approach to feature selection, which is based on statistical significant test, in contrast to the conventional methods, which use sub-structure size.

To simplify the discussion, we restrict ourselves to dealing hereafter with the two-class (positive and negative) supervised classification problem. In our approach, we test the statistical deviation of all sub-structures in the training samples between the appearance of positive samples and negative samples, and then, select only the sub-structures which are larger than a certain threshold $\tau$ as features. This allows us to select only the statistically significant sub-structures. In this paper, we explains our proposed method by using the chi-squared ($\chi^2$) value as a statistical metric. We note, however, we can use many types of statistical metrics in our proposed method.

Table 1: Contingency table and notation for the chi-squared value

|  | $c$ | $\bar{c}$ | $\sum$ row |
|---|---|---|---|
| $u$ | $O_{uc}$ | $O_{u\bar{c}}$ | $O_u$ |
| $\bar{u}$ | $O_{\bar{u}c}$ | $O_{\bar{u}\bar{c}}$ | $O_{\bar{u}}$ |
| $\sum$ column | $O_c$ | $O_{\bar{c}}$ | $N$ |

First, we briefly explain how to calculate the $\chi^2$ value by referring to Table 1. $c$ and $\bar{c}$ represent the names of classes, $c$ for the positive class and $\bar{c}$ for the negative class. $O_{ij}$, where $i \in \{u, \bar{u}\}$ and $j \in \{c, \bar{c}\}$, rep-

resents the number of samples in each case. $O_{u\bar{c}}$, for instance, represents the number of $u$ that appeared in $\bar{c}$. Let $N$ be the total number of training samples. Since $N$ and $O_c$ are constant for training samples, $\chi^2$ can be obtained as a function of $O_u$ and $O_{uc}$. The $\chi^2$ value expresses the normalized deviation of the observation from the expectation: $\text{chi}(O_u, O_{uc}) = \sum_{i \in \{u, \bar{u}\}, j \in \{c, \bar{c}\}} (O_{ij} - E_{ij})^2 / E_{ij}$, where $E_{ij} = n \cdot O_i / n \cdot O_j / n$, which represents the expectation. We simply represent $\text{chi}(O_u, O_{uc})$ as $\chi^2(u)$.

In the kernel calculation with the statistical feature selection, if $\chi^2(u) < \tau$ holds, that is, $u$ is not statistically significant, then $u$ is eliminated from the features, and the value of $u$ is presumed to be 0 for the kernel value. Therefore, the sequence and tree kernel with feature selection (SK,TK+FS) can be defined as follows:

$$K^{\text{SK,TK+FS}}(T^1, T^2) = \sum_{u \in \{u | \tau \leq \chi^2(u), u \in P(\mathcal{L}_1^n)\}} \sum_{t^1 \in \text{sub}(T^1)} C_u(t^1)^{\gamma_u(t^1)} \sum_{t^2 \in \text{sub}(T^2)} C_u(t^2)^{\gamma_u(t^2)}.$$

(2)

The difference with their original kernels is simply the condition of the first summation, which is $\tau \leq \chi^2(u)$.

The basic idea of using a statistical metric to select features is quite natural, but it is not a very attractive approach. We note, however, it is not clear how to calculate that kernels efficiently with a statistical feature selection. It is computationally infeasible to calculate $\chi^2(u)$ for all possible $u$ with a naive exhaustive method. In our approach, we take advantage of *sub-structure mining* algorithms in order to calculate $\chi^2(u)$ efficiently and to embed statistical feature selection to the kernel calculation. Formally, sub-structure mining is to find the complete set, but no-duplication, of all significant (generally frequent) sub-structures from dataset. Specifically, we apply combination of a sequential pattern mining technique, PrefixSpan [9], and a statistical metric pruning (SMP) method, Apriori SMP [8]. PrefixSpan can substantially reduce the search space of enumerating all significant sub-sequences. Briefly saying, it finds any sub-sequences $uw$ whose size is $n$, by searching a single symbol $w$ in the projected database of the sub-sequence (prefix) $u$ of size $n-1$. The projected database is a partial database which only contains all postfixes (pointers in the implementation) of appeared the prefix $u$ in the database. It starts searching from $n = 1$, that is, it enumerates all the significant sub-sequences by the recursive calculation of *pattern-growth*, searching in the projected database of prefix $u$ and adding a symbol $w$ to $u$, and *prefix-projection*, making projected database of $uw$.

Before explaining the algorithm of the proposed kernels, we introduce the upper bound of the $\chi^2$ value. The upper bound of the $\chi^2$ value of a sequence $uv$, which is the concatenation of sequences $u$ and $v$, can be calculated by the value of the contingency table of the prefix $u$ [8]: $\chi^2(uv) \leq \widehat{\chi}^2(u) = \max\left(\text{chi}(O_{uc}, O_{uc}), \text{chi}(O_u - O_{uc}, 0)\right)$. This upper bound indicates that if $\widehat{\chi}^2(u) < \tau$ holds, no (super-)sequences $uv$, whose prefix is $u$, can be larger than threshold, $\tau \leq \chi^2(uv)$. In our context, we can eliminate all (super-)sequences $uv$ from candidates of the feature without the explicit evaluation of $uv$.

Using this property in the PrefixSpan algorithm, we can eliminate to evaluate all the (super-)sequences $uv$ by evaluating the upper bound of sequence $u$. After finding the number of individual symbol $w$ appeared in projected database of $u$, we evaluate $uw$ in the following three conditions: (1) $\tau \leq \chi^2(uw)$, (2) $\tau > \chi^2(uw)$, $\tau > \widehat{\chi}^2(uw)$, and (3) $\tau > \chi^2(uw)$, $\tau \leq \widehat{\chi}^2(uw)$. With condition (1), sub-sequence $uw$ is selected as the feature. With condition (2), $uw$ is pruned, that is, all $uwv$ are also pruned from search space. With condition (3), $uw$ is not a significant, however, $uwv$ can be a significant; thus $uw$ is not selected as features, however, mining is continue to $uwv$. Figure 1 shows an example of searching and pruning the sub-sequences to select significant features by the PrefixSpan with SMP algorithm.

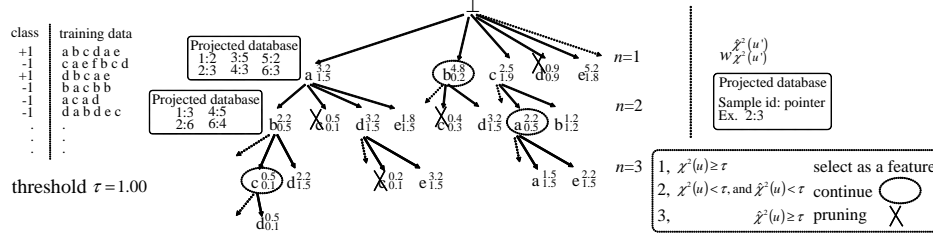

Figure 1: Example of searching and pruning the sub-sequences by PrefixSpan with SMP algorithm

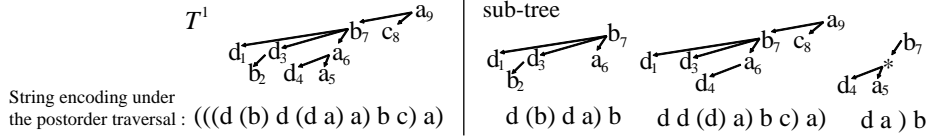

Figure 2: Example of the string encoding for trees under the postorder traversal

The famous tree mining algorithm [12] cannot be simply applied as a feature selection method for the proposed tree kernels, because this tree mining executes preorder search of trees while tree kernels calculate the kernel in postorder. Thus, we take advantage of the *string (sequence) encoding* method for trees and treat them in sequence kernels. Figure 2 shows an example of the string encoding for trees under the postorder traversal. The brackets indicate the hierarchical relation between their left and right hand side nodes. We treat these brackets as a special symbol during the sequential pattern mining phase. Sub-trees are evaluated as the same if and only if the string encoded sub-sequences are exactly the same including brackets. For example, 'd ) b ) a' and 'd b ) a' are different.

We previously said that sequence can be treated as one of trees. We also encode in the case of sequence; for example a sequence 'a b c d' is encoded in '((((a) b) c) d)'. That is, we can define sequence and tree kernels with our feature selection method in the same form.

**Sequence and Tree Kernels with Statistical Feature Mining:** Sequence and Tree kernels with our proposed feature selection method is defined in the following equations.

$$K^{\text{SK,TK+FS}}(T^1, T^2; \mathcal{D}) = \sum_{1 \leq i \leq |T^1|} \sum_{1 \leq j \leq |T^2|} \mathcal{H}_n(T_i^1, T_j^2; \mathcal{D}) \qquad (3)$$

$\mathcal{D}$ represents the training data, and $i$ and $j$ represent indices of nods in postorder of $T^1$ and $T^2$, respectively. Let $\mathcal{H}_n(T_i^1, T_j^2; \mathcal{D})$ be a function that returns the sum value of all statistically significant common sub-sequences $u$ if $t_i^1 = t_j^2$ and $|u| \leq n$.

$$\mathcal{H}_n(T_i^1, T_j^2; \mathcal{D}) = \sum_{u \in \Gamma_n(T_i^1, T_j^2; \mathcal{D})} \mathcal{J}_u(T_i^1, T_j^2; \mathcal{D}), \qquad (4)$$

where $\Gamma_n(T_i^1, T_j^2; \mathcal{D})$ represents a set of sub-sequences, which is $|u| \leq n$, that satisfy the above condition 1. Then, let $\mathcal{J}_u(T_i^1, T_j^2; \mathcal{D})$, $\mathcal{J}_u'(T_i^1, T_j^2; \mathcal{D})$ and $\mathcal{J}_u''(T_i^1, T_j^2; \mathcal{D})$ be functions that calculate the value of the common sub-sequences between $T_i^1$ and $T_j^2$ recursively.

$$\mathcal{J}_{uw}(T_i^1, T_j^2) = \begin{cases} \mathcal{J}_u'(T_i^1, T_j^2; \mathcal{D}) \cdot \mathcal{I}_w(t_i^1, t_j^2) & \text{if } uw \in \widehat{\Gamma}_n(T_i^1, T_j^2; \mathcal{D}), \\ 0 & \text{otherwise,} \end{cases} \qquad (5)$$

where $\mathcal{I}_w(t_i^1, t_j^2)$ is a function that returns 1 iff $t_i^1 = w$ and $t_j^2 = w$, and 0 otherwise. $\widehat{\Gamma}_n(T_i^1, T_j^2; \mathcal{D})$ is a set of sub-sequences, which is $|u| \leq n$, that satisfy condition (3). We introduce a special symbol $\Lambda$ to represent an "empty sequence", and define $\Lambda w = w$ and $|\Lambda w| = 1$.

$$\mathcal{J}_u'(T_i^1, T_j^2; \mathcal{D}) = \begin{cases} 1 & \text{if} \quad u = \Lambda, \\ 0 & \text{if} \quad j = 0 \quad \text{and} \quad u \neq \Lambda, \\ \lambda \mathcal{J}_u'(T_i^1, T_{j-1}^2; \mathcal{D}) + \mathcal{J}_u''(T_i^1, T_{j-1}^2, \mathcal{D}) & \text{otherwise,} \end{cases} \quad (6)$$

$$\mathcal{J}_u''(T_i^1, T_j^2; \mathcal{D}) = \begin{cases} 0 & \text{if} \quad i = 0, \\ \lambda \mathcal{J}_u''(T_{i-1}^1, T_j^2; \mathcal{D}) + \mathcal{J}_u(T_{i-1}^1, T_j^2; \mathcal{D}) & \text{otherwise.} \end{cases} \quad (7)$$

The following equations are introduced to select a set of significant sub-sequences.

$$\Gamma_n(T_i^1, T_j^2; \mathcal{D}) = \{u \mid u \in \widehat{\Gamma}_n(T_i^1, T_j^2; \mathcal{D}), \tau \leq \chi^2(u), u_{|u|} \in \cap_{i=1}^{|u|-1} \text{ans}(u_i)\} \quad (8)$$

$u_{|u|} \in \cap_{i=1}^{|u|-1} \text{ans}(u_i)$ evaluates if a sub-sequence $u$ is complete sub-tree, where $\text{ans}(u_i)$ returns ancestor of the node $u_i$. For example, 'd ) b a' is not a complete subtree, because the last node 'a' is not an ancestor of 'd' and 'b'.

$$\widehat{\Gamma}_n(T_i^1, T_j^2; \mathcal{D}) = \begin{cases} \Psi_n(\widehat{\Gamma}_n'(T_i^1, T_j^2; \mathcal{D}), t_i^1) \cup \{t_i^1\} & \text{if} \quad t_i^1 = t_j^2, \\ \emptyset & \text{otherwise,} \end{cases} \quad (9)$$

where $\Psi_n(F, w) = \{uw \mid u \in F, \tau \leq \widehat{\chi}^2(uw), |uw| \leq n\}$, and $F$ represents a set of sub-sequences. Note that $\Gamma_n(T_i^1, T_j^2; \mathcal{D})$ and $\widehat{\Gamma}_n(T_i^1, T_j^2; \mathcal{D})$ have only sub-sequences $u$ that satisfy $\tau \leq \chi^2(uw)$ and $\tau \leq \widehat{\chi}^2(uw)$, respectively, iff $t_i^1 = t_j^2$ and $|uw| \leq n$; otherwise they become empty sets.

The following two equations are introduced for recursive the set operation to calculate $\Gamma_n(T_i^1, T_j^2; \mathcal{D})$ and $\widehat{\Gamma}_n(T_i^1, T_j^2; \mathcal{D})$.

$$\widehat{\Gamma}_n'(T_i^1, T_j^2; \mathcal{D}) = \begin{cases} \emptyset & \text{if } j = 0, \\ \widehat{\Gamma}_n'(T_i^1, T_{j-1}^2; \mathcal{D}) \cup \widehat{\Gamma}_n''(T_i^1, T_{j-1}^2; \mathcal{D}) & \text{otherwise,} \end{cases} \quad (10)$$

$$\widehat{\Gamma}_n''(T_i^1, T_j^2; \mathcal{D}) = \begin{cases} \emptyset & \text{if } i = 0 , \\ \widehat{\Gamma}_n''(T_{i-1}^1, T_j^2; \mathcal{D}) \cup \widehat{\Gamma}_n(T_{i-1}^1, T_j^2; \mathcal{D}) & \text{otherwise.} \end{cases} \quad (11)$$

In the implementation, $\chi^2(uw)$ and $\widehat{\chi}^2(uw)$, where $uw$ represents a concatenation of a sequence $u$ and a symbol $w$, can be calculated by a set of pointers of $u$ against data and the number of appearance of $w$ in backside of the pointers. We note that the set of pointers of $uw$ can be simply obtained from previous search of $u$. With condition (1), $uw$ is stored in $\Gamma_n$ and $\widehat{\Gamma}_n$. With condition (3), $uw$ is only stored in $\widehat{\Gamma}_n$.

There are some technique in order to calculate kernel faster in the implementation. For example, since $\chi^2(u)$ and $\hat{\chi}^2(u)$ are constant against the same data, we only have to calculate them once. We store the internal search results of PrefixSpan with SMP algorithm in a TRIE structure. After that, we look in that results in TRIE instead of explicitly calculate $\chi^2(u)$ again when the kernel finds the same sub-sequence. Moreover, when the projected database is exactly the same, these sub-sequences can be merged since the value of $\chi^2(uv)$ and $\hat{\chi}^2(uv)$ for any postfix $v$ are exactly the same. Moreover, we introduce a 'transposed index' for fast evaluation of $\chi^2(u)$ and $\hat{\chi}^2(u)$. By using that, we only have to look up that index of $w$ to evaluate whether or not any $uw$ are significant features.

Equations (4) to (7) can be performed in the same as the original DP based kernel calculation. The recursive set operations of Equations (9) to (11) can be executed as well as

Table 2: Experimental Results

| | Question Classification | | | | | Subjectivity Detection | | | | | Polarity Identification | | | | |
|---|---|---|---|---|---|---|---|---|---|---|---|---|---|---|---|
| $n$ | 1 | 2 | 3 | 4 | $\infty$ | 1 | 2 | 3 | 4 | $\infty$ | 1 | 2 | 3 | 4 | $\infty$ |
| **SK+FS** | - | .823 | .827 | .824 | .822 | - | .822 | .839 | .841 | .842 | - | .824 | .838 | .839 | .839 |
| SK | - | .808 | .818 | .808 | .797 | - | .823 | .824 | .809 | .772 | - | .835 | .835 | .833 | .789 |
| **TK+FS** | - | .812 | .815 | .812 | .812 | - | .834 | .857 | .854 | .856 | - | .830 | .832 | .835 | .833 |
| TK | - | .802 | .802 | .797 | .783 | - | .842 | .850 | .830 | .755 | - | .828 | .827 | .820 | .745 |
| BOW-K | .754 | .792 | .790 | .778 | - | .717 | 729 | .715 | .649 | - | .740 | .810 | .822 | .795 | - |

Equations (5) to (7). Moreover, calculating $\chi^2(u)$ and $\hat{\chi}^2(u)$ with sub-structure mining algorithms allow to calculate the same order of the DP based kernel calculation. As a result, statistical feature selection can be embedded in original kernel calculation based on the DP.

Essentially, the worst case time complexity of the proposed method will become exponential, since we enumerate individual sub-structures in sub-structure mining phase. However, actual calculation time in the most cases of our experiments is even faster than original kernel calculation, since search space pruning efficiently remove vain calculation and the implementation techniques briefly explained above provide practical calculation speed.

We note that if we set $\tau = 0$, which means all features are dealt with kernel calculation, we can get exactly the same kernel value as the original tree kernel.

## 5   Experiments and Results

We evaluated the performance of the proposed method in actual NLP tasks, namely *English question classification* (EQC), *subjectivity detection* (SD) and *polarity identification* (PI) tasks. These tasks are defined as a text categorization task: it maps a given sentence into one of the pre-defined classes. We used data provided by [6] for EQC, that contains about 5500 questions with 50 question types. SD data was created from Mainichi news articles, and the size was 2095 sentences consisting of 822 subjective sentences. PI data has 5564 sentences with 2671 positive opinion. By using these data, we compared the proposed method (SK+FS and TK+FS) with a conventional method (SK or TK), as discussed in Section 3, and with *bag-of-words* (BOW) Kernel (BOW-K)[4] as baseline methods. We used word sequences for input objects of sequence kernels and word dependency trees for tree kernels.

Support Vector Machine (SVM) was selected as the kernel-based classifier for training and classification with a soft margin parameter $C = 1000$. We used the *one-vs-rest* classifier of SVM as the multi-class classification method for EQC. We evaluated the performance with label accuracy by using ten-fold cross validation: eight for training, one for development and remaining one for test set. The parameter $\lambda$ and $\tau$ was automatically selected from the value set of $\lambda = \{0.1, 0.3, 0.5, 0.7, 0.9\}$ and $\tau = \{3.84, 6.63\}$ by the development test. Note that these two values represent the 10% and 5% levels of significance in the $\chi^2$ distribution with one degree of freedom, which used the $\chi^2$ significant test.

Tables 2 shows our experimental results. where $n$ in each table indicates the restriction of the sub-structure size, and $n = \infty$ means all possible sub-structures are used. As shown in this table, SK or TK achieve maximum performance when $n = 2$ or 3. The performance deteriorates considerably once $n$ exceeds 4 or more. This implies that larger sub-structures degrade classification performance, which showed the same tendency as in the previous studies discussed in Section 3. This is evidence of over-fitting in learning. On the other hand, SK+FS and TK+FS provided consistently better performance than the conventional methods. Moreover, the experiments confirmed one important fact: in some cases, maximum performance was achieved with $n = \infty$. This indicates that certain sub-sequences

created using very large structures can be extremely effective. If the performance is improved by using a larger $n$, this means that significant features do exist. Thus, we can improve the performance of some classification problems by dealing with larger substructures. Even if optimum performance was not achieved with $n = \infty$, the difference from the performance of a smaller $n$ is quite small compared to that of SK and TK. This indicates that our method is very robust against sub-structure size.

# 6   Conclusions

This paper proposed a statistical feature selection method for sequence kernels and tree kernels. Our approach can select significant features automatically based on a statistical significance test. The proposed method can be embedded in the original DP based kernel calculation process by using sub-structure mining algorithms.

Our experiments demonstrated that our method is superior to conventional methods. Moreover, the results indicate that complex features exist and can be effective. Our method can employ them without over-fitting problems, which yields benefits in terms of concept and performance.

# References

[1] N. Cancedda, E. Gaussier, C. Goutte, and J.-M. Renders. Word-Sequence Kernels. *Journal of Machine Learning Research*, 3:1059–1082, 2003.

[2] M. Collins and N. Duffy. Convolution kernels for natural language. In *Proc. of Neural Information Processing Systems (NIPS'2001)*, 2001.

[3] D. Haussler. Convolution kernels on discrete structures. In *Technical Report UCS-CRL-99-10*. UC Santa Cruz, 1999.

[4] T. Joachims. Text Categorization with Support Vector Machines: Learning with Many Relevant Features. In *Proc. of European Conference on Machine Learning (ECML '98)*, pages 137–142, 1998.

[5] H. Kashima and T. Koyanagi. Kernels for Semi-Structured Data. In *Proc. 19th International Conference on Machine Learning (ICML2002)*, pages 291–298, 2002.

[6] X. Li and D. Roth. Learning Question Classifiers. In *Proc. of the 19th International Conference on Computational Linguistics (COLING 2002)*, pages 556–562, 2002.

[7] H. Lodhi, C. Saunders, J. Shawe-Taylor, N. Cristianini, and C. Watkins. Text Classification Using String Kernel. *Journal of Machine Learning Research*, 2:419–444, 2002.

[8] S. Morishita and J. Sese. Traversing Itemset Lattices with Statistical Metric Pruning. In *Proc. of ACM SIGACT-SIGMOD-SIGART Symp. on Database Systems (PODS'00)*, pages 226–236, 2000.

[9] J. Pei, J. Han, B. Mortazavi-Asl, and H. Pinto. PrefixSpan: Mining Sequential Patterns Efficiently by Prefix-Projected Pattern Growth. In *Proc. of the 17th International Conference on Data Engineering (ICDE 2001)*, pages 215–224, 2001.

[10] J. Suzuki, Y. Sasaki, and E. Maeda. Kernels for Structured Natural Language Data. In *Proc. of the 17th Annual Conference on Neural Information Processing Systems (NIPS2003)*, 2003.

[11] C. Watkins. Dynamic alignment kernels. In *Technical Report CSD-TR-98-11*. Royal Holloway, University of London Computer Science Department, 1999.

[12] M. J. Zaki. Efficiently Mining Frequent Trees in a Forest. In *Proc. of the 8th International Conference on Knowledge Discovery and Data Mining (KDD'02)*, pages 71–80, 2002.
